# Using Free Energies to Represent Q-values in a Multiagent Reinforcement Learning Task

**Brian Sallans**
Department of Computer Science
University of Toronto
Toronto  M5S 2Z9  Canada
*sallans@cs.toronto.edu*

**Geoffrey E. Hinton**
Gatsby Computational Neuroscience Unit
University College London
London  WC1N 3AR  U.K.
*hinton@gatsby.ucl.ac.uk*

## Abstract

The problem of reinforcement learning in large factored Markov decision processes is explored. The Q-value of a state-action pair is approximated by the free energy of a product of experts network. Network parameters are learned on-line using a modified SARSA algorithm which minimizes the inconsistency of the Q-values of consecutive state-action pairs. Actions are chosen based on the current value estimates by fixing the current state and sampling actions from the network using Gibbs sampling. The algorithm is tested on a co-operative multi-agent task. The product of experts model is found to perform comparably to table-based Q-learning for small instances of the task, and continues to perform well when the problem becomes too large for a table-based representation.

## 1  Introduction

Online Reinforcement Learning (RL) algorithms try to find a policy which maximizes the expected time-discounted reward provided by the environment. They do this by performing sample backups to learn a value function over states or state-action pairs [1]. If the decision problem is Markov in the observed states, then the optimal value function over state-action pairs (the Q-function) yields all of the information required to find the optimal policy for the decision problem. For example, when the Q-function is represented as a table, the optimal action for a given state can be found simply by searching the row of the table corresponding to that state.

### 1.1  Factored Markov Decision Processes

In many cases the dimensionality of the problem makes a table representation impractical, so a more compact representation that makes use of the structure inherent in the problem is required. In a co-operative multi-agent system, for example, it is natural to represent both the state and action as sets of variables (one for each agent). We expect that the mapping from the combined states of all the agents to the combined actions of all the agents is not arbitrary: Given an individual agent's state, that agent's action might be largely independent of the other agents' exact states and actions, at least for some regions of the combined state space. We expect that a *factored* representation of the Q-value function will be appropriate

for two reasons: The original representation of the combined states and combined actions is factored, and the ways in which the optimal actions of one agent are dependent on the states and actions of other agents might be well captured by a small number of "hidden" factors rather than the exponential number required to express arbitrary mappings.

## 1.2 Actor-Critic Architectures

If a non-linear function approximator is used to model the Q-function, then it is difficult and time consuming to extract the policy directly from the Q-function because a non-linear optimization must be solved for each action choice. One solution, called an *actor-critic* architecture, is to use a separate function approximator to model the policy (i.e. to approximate the non-linear optimization) [2, 3]. This has the advantage of being fast, and allows us to explicitly learn a stochastic policy, which can be advantageous if the underlying problem is not strictly Markov [4]. However, a specific parameterized family of policies must be chosen a priori.

Instead we present a method where the Q-value of a state-action pair is represented (up to an additive constant) by the negative free-energy, $-F$, of the state-action pair under a non-causal graphical model. The graphical model is a product of experts [5] which has two very useful properties: Given a state-action pair, the exact free energy is easily computed, and the derivative of this free energy *w.r.t.* each parameter of the network is also very simple. The model is trained to minimize the inconsistency between the free-energy of a state-action pair and the discounted free energy of the next state-action pair, taking into account the immediate reinforcement. After training, a good action for a given state can be found by clamping the state and drawing a sample of the action variables using Gibbs sampling [6]. Although finding optimal actions would still be difficult for large problems, selecting an action with a probability that is approximately proportional to $\exp(-F)$ can be done with a modest number of iterations of Gibbs sampling.

## 1.3 Markov Decision Processes

We will concentrate on finite, factored, Markov decision processes (factored MDPs), in which each state and action is represented as a set of discrete variables. Formally, a factored MDP consists of the set $\{\{\mathcal{S}_\alpha\}_{\alpha=1}^M, \{\mathcal{A}_\beta\}_{\beta=1}^N, \{s_\alpha^0\}_{\alpha=1}^M, P, P_r\}$, where: $\mathcal{S}_\alpha$ is the set of possible values for state variable $\alpha$; $\mathcal{A}_\beta$ is the set of possible values for action variable $\beta$; $s_\alpha^0$ is the initial value for state variable $\alpha$; $P$ is a transition distribution $P(\mathbf{s}^{t+1}|\mathbf{s}^t, \mathbf{a}^t)$; and $P_r$ is a reward distribution $P(r^t|\mathbf{s}^t, \mathbf{a}^t, \mathbf{s}^{t+1})$. A state is an $M$-tuple and an action is an $N$-tuple.

The goal of solving an MDP is to find a *policy*, which is a sequence of (possibly stochastic) mappings $\pi^t : \mathcal{S}_1 \times \mathcal{S}_2 \times ... \times \mathcal{S}_M \rightarrow \mathcal{A}_1 \times \mathcal{A}_2 \times ... \times \mathcal{A}_N$ which maximize the total expected reward received over the course of the task:

$$\langle R^t \rangle_{\pi^t} = \langle r^t + \gamma r^{t+1} + ... + \gamma^{T-t} r^T \rangle_{\pi^t} \tag{1}$$

where $\gamma$ is a discount factor and $\langle \cdot \rangle_{\pi^t}$ denotes the expectation taken with respect to policy $\pi^t$. We will focus on the case when the policy is stationary: $\pi^t$ is identical for all $t$.

## 2 Approximating Q-values with a Product of Experts

As the number of state and action variables increases, a table representation quickly becomes intractable. We represent the value of a state and action as the negative free-energy (up to a constant) under a product of experts model (see Figure 1(a)).

With a product of experts, the probability assigned to a state-action pair, $(\mathbf{s}, \mathbf{a})$ is just the (normalized) product of the probabilities assigned to $(\mathbf{s}, \mathbf{a})$ under each of the individual

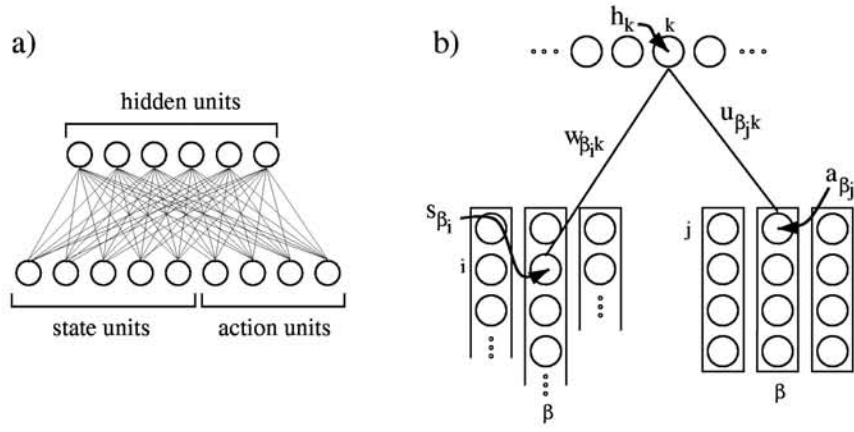

Figure 1: a) The Boltzmann product of experts. The estimated Q-value (up to an additive constant) of a setting of the state and action units is found by holding these units fixed and computing the free energy of the network. Actions are selected by alternating between updating all of the hidden units in parallel and updating all of the action units in parallel, with the state units held constant. b) A multinomial state or action variable is represented by a set of "one-of-$n$" binary units in which exactly one is on.

experts:

$$p(\mathbf{s}, \mathbf{a}|\theta_1, ..., \theta_K) = \frac{\prod_{k=1}^{K} p_k(\mathbf{s}, \mathbf{a}|\theta_k)}{\sum_{(\mathbf{s}', \mathbf{a}')} \prod_k p_k(\mathbf{s}', \mathbf{a}'|\theta_k)} \tag{2}$$

where $\{\theta_1, ..., \theta_K\}$ are parameters of the $K$ experts and $(\mathbf{s}', \mathbf{a}')$ indexes all possible state-action pairs.

In the following, we will assume that there are an equal number of state and action variables (i.e. $M = N$); and that each state or action variable has the same arity ($\forall \alpha, \beta, |\mathcal{S}^\alpha| = |\mathcal{S}^\beta|$ and $|\mathcal{A}^\alpha| = |\mathcal{A}^\beta|$). These assumptions are appropriate, for example, when there is one state and action variable for each agent in a multi-agent task. Extension to the general case is straight forward. In the following, $\beta$ will index agents.

Many kinds of "experts" could be used while still retaining the useful properties of the PoE. We will focus on the case where each expert is a single binary sigmoid unit because it is particularly suited to the discrete tasks we consider here. Each agent's (multinomial) state or action is represented using a "one-of-$N$" set of binary units which are constrained so that exactly one of them is on. The product of experts is then a bipartite "Restricted Boltzmann Machine" [5]. We use $s_{\beta_i}$ to denote agent $\beta$'s $i^{th}$ state and $a_{\beta_j}$ to denote its $j^{th}$ action. We will denote the binary latent variables of the "experts" by $h_k$ (see Figure 1(b)).

For a state $\mathbf{s} = \{s_{\beta_i}\}$ and an action $\mathbf{a} = \{a_{\beta_j}\}$, the free energy is given by the expected energy given the posterior distribution of the hidden units minus the entropy of this posterior distribution. This is simple to compute because the hidden units are independent in the posterior distribution:

$$F(\mathbf{s}, \mathbf{a}) = -\sum_{k=1}^{K} \sum_{\beta=1}^{M} \left( \sum_{i=1}^{|\mathcal{S}|} (w_{\beta_i k} s_{\beta_i} \widehat{h}_k + b_{\beta_i} s_{\beta_i}) + \sum_{j=1}^{|\mathcal{A}|} (u_{\beta_j k} a_{\beta_j} \widehat{h}_k + b_{\beta_j} a_{\beta_j}) \right)$$
$$- \sum_{k=1}^{K} b_k \widehat{h}_k + \sum_{k=1}^{K} \widehat{h}_k \log \widehat{h}_k + (1 - \widehat{h}_k) \log(1 - \widehat{h}_k) - C_F \tag{3}$$

where $w_{\beta_i k}$ is the weight from the $k^{\text{th}}$ expert to binary state variable $s_{\beta_i}$; $u_{\beta_j k}$ is the weight from the $k^{\text{th}}$ expert to binary action variable $a_{\beta_j}$; $b_k$, $b_{\beta_i}$ and $b_{\beta_j}$ are biases; and

$$\widehat{h}_k = \sigma \left\{ \sum_{\beta=1}^{M} \left( \sum_{i=1}^{|S|} w_{\beta_i k} s_{\beta_i k} + \sum_{j=1}^{|A|} u_{\beta_j k} a_{\beta_j k} \right) + b_k \right\} \tag{4}$$

is the expected value of each expert given the data where $\sigma(x) = 1/1 + e^{-x}$ denotes the logistic function. $C_F$ is an additive constant equal to the log of the partition function. The first two terms of (3) corresponds to an unnormalized negative log-likelihood, and the third to the negative entropy of the distribution over the hidden units given the data. The free energy can be computed tractably because inference is tractable in a product of experts: under the product model each expert is independent of the others given the data. We can efficiently compute the exact free energy of a state and action under the product model, up to an additive constant. The Q-function will be approximated by the negative free-energy (or goodness), without the constant:

$$Q(\mathbf{s}, \mathbf{a}) \approx -F(\mathbf{s}, \mathbf{a}) + C_F \tag{5}$$

## 2.1 Learning the Parameters

The parameters of the model must be adjusted so that the goodness of a state-action under the product model approximates its actual Q-value. This is done with a modified SARSA learning rule designed to minimize the Bellman error [7, 8]. If we consider a delta-rule update where the target for input $(\mathbf{s}^t, \mathbf{a}^t)$ is $r^t + \gamma Q(\mathbf{s}^{t+1}, \mathbf{a}^{t+1})$, then (for example) the update for $w_{\beta_i k}$ is given by:

$$\Delta w_{\beta_i k} \propto \left( r^t + \gamma Q(\mathbf{s}^{t+1}, \mathbf{a}^{t+1}) - Q(\mathbf{s}^t, \mathbf{a}^t) \right) \frac{\partial Q(\mathbf{s}^t, \mathbf{a}^t)}{\partial w_{\beta_i k}} \tag{6}$$

$$\frac{\partial Q(\mathbf{s}^t, \mathbf{a}^t)}{\partial w_{\beta_i k}} = \widehat{h}_k^t s_{\beta_i}^t \tag{7}$$

The other weights and biases are updated similarly. Although there is no proof of convergence for this learning rule, it works well in practice even though it ignores the effect of changes in $w_{\beta_i k}$ on $Q(\mathbf{s}^{t+1}, \mathbf{a}^{t+1})$.

## 2.2 Sampling Actions

Given a trained network and the current state $\mathbf{s}^t$, we need to generate actions according to their goodness. We would like to select actions according to a Boltzmann exploration scheme in which the probability of selecting an action is proportional to $e^{Q/T}$. This selection scheme has the desirable property that it optimizes the trade-off between the expected payoff, $Q$, and the entropy of the selection distribution, where $T$ is the relative importance of exploration versus exploitation. Fortunately, the additive constant, $C_F$, does not need to be known in order to select actions in this way. It is sufficient to do alternating Gibbs sampling. We start with an arbitrary initial action represented on the action units. Holding the state units fixed we update all of the hidden units in parallel so that we get a sample from the posterior distribution over the hidden units given the state and the action. Then we update all of the action units in parallel so that we get a sample from the posterior distribution over actions given the states of the hidden units. When updating the states of the action units, we use a "softmax" to enforce the one-of-N constraint within a set of binary units that represent mutually exclusive actions of the same agent. When the alternating Gibbs sampling reaches equilibrium it draws unbiased samples of actions according to their Q-value. For the networks we used, 50 Gibbs iterations appeared to be sufficient to come close to the equilibrium distribution.

# 3   Experimental Results

To test the algorithm we introduce a co-operative multi-agent task in which there are offensive players trying to reach an end-zone, and defensive players trying to block them (see Figure 2).

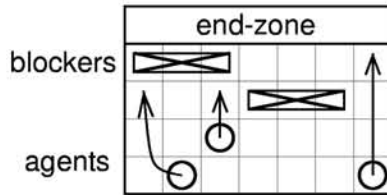

Figure 2: An example of the "blocker" task. Agents must get past the blockers to the end-zone. The blockers are pre-programmed with a strategy to stop them, but if they co-operate the blockers cannot stop them all simultaneously.

The task is co-operative: As long as one agent reaches the end-zone, the "team" is rewarded. The team receives a reward of $+1$ when an agent reaches the end-zone, and a reward of $-1$ otherwise. The blockers are pre-programmed with a fixed blocking strategy. Each agent occupies one square on the grid, and each blocker occupies three horizontally adjacent squares. An agent cannot move into a square occupied by a blocker or another agent. The task has non-wrap-around edge conditions on the east, west and south sides of the field, and the blockers and agents can move north, south, east or west.

A product of experts (PoE) network with 4 hidden units was trained on a $5 \times 4$ blocker task with two agents and one blocker. The combined state consisted of three position variables (two agents and one blocker) which could take on integer values $\{1, ..., 20\}$. The combined action consisted of two action variables taking on values from $\{1, ..., 4\}$.

The network was run twice, once for 60 000 combined actions and once for 400 000 combined actions, with a learning rate going from 0.1 to 0.01 linearly and temperature going from 1.0 to 0.01 exponentially over the course of training. Each trial was terminated after either the end-zone was reached, or 20 combined actions were taken, whichever occurred first. Each trial was initialized with the blocker placed randomly in the top row and the agents placed randomly in the bottom row. The same learning rate and temperature schedule were used to train a Q-learner with a table containing 128,000 elements ($20^3 \times 4^2$), except that the Q-learner was allowed to train for 1 million combined actions. After training each policy was run for 10,000 steps, and all rewards were totaled. The two algorithms were also compared to a hand-coded policy, where the agents first move to opposite sides of the field and then move to the end-zone. In this case, all of the algorithms performed comparably, and the POE network performing well even for a short training time.

A PoE network with 16 hidden units was trained on a $4 \times 7$ blockers task with three agents and two blockers. Again, the input consisted of position variables for each blocker and agent, and and action variables for each agent. The network was trained for 400 000 combined actions, with the a learning rate from 0.01 to 0.001 and the same temperature schedule as the previous task. Each trial was terminated after either the end-zone was reached, or 40 steps were taken, whichever occurred first. After training, the resultant policy was run for 10,000 steps and the rewards received were totaled. As the table representation would have over a billion elements ($28^5 \times 4^3$), a table based Q-learner could not be trained for comparison. The hand-coded policy moved agents 1, 2 and 3 to the left, middle and right column respectively, and then moved all agents towards the end-zone. The PoE performed comparably to this hand-coded policy. The results for all experiments are summarized in Table 1.

Table 1: Experimental Results

| Algorithm | Reward |
|---|---|
| Random policy (5 × 4, 2 agents, 1 blocker) | -9986 |
| hand-coded (5 × 4, 2 agents, 1 blocker) | -6782 |
| Q-learning (5 × 4, 2 agents, 1 blocker, 1000K steps) | -6904 |
| PoE (5 × 4, 2 agents, 1 blocker, 60K steps) | -7303 |
| PoE (5 × 4, 2 agents, 1 blocker, 400K steps) | -6738 |
| Random policy (4 × 7, 3 agents, 2 blockers) | -9486 |
| hand-coded (4 × 7, 3 agents, 2 blockers) | -7074 |
| PoE (4 × 7, 3 agents, 2 blockers, 400K steps) | -7631 |

## 4  Discussion

Each hidden unit in the product model implements a probabilistic constraint that captures one aspect of the relationship between combined states and combined actions in a good policy. In practice the hidden units tend to represent particular strategies that are relevant in particular parts of the combined state space. This suggests that the hidden units could be used for hierarchical or temporal learning. A reinforcement learner could, for example, learn the dynamics between hidden unit values (useful for POMDPs) and the rewards associated with hidden unit activations.

Because the PoE network implicitly represents a joint probability distribution over state-action pairs, it can be queried in ways that are not normally possible for an actor network. Given any subset of state and action variables, the remainder can be sampled from the network using Gibbs sampling. This makes it easy to answer questions of the form: "How should agent 3 behave given fixed actions for agents 1 and 2?" or "I can see some of the state variables but not others. What values would I most like to see for the others?". Further, because there is an efficient unsupervised learning algorithm for PoE networks, an agent could improve its policy by watching another agent's actions and making them more probable under its own model.

There are a number of related works, both in the fields of reinforcement learning and unsupervised learning. The SARSA algorithm is from [7, 8]. A delta-rule update similar to ours was explored by [9] for POMDPs and Q-learning. Factored MDPs and function approximators have a long history in the adaptive control and RL literature (see for example [10]).

Our method is also closely related to actor-critic methods [2, 3]. Normally with an actor-critic method, the actor network can be viewed as a biased scheme for selecting actions according to the value assigned by the critic. The selection is biased by the choice of parameterization. Our method of action selection is unbiased (if the Markov chain is allowed to converge). Further, the resultant policy can potentially be much more complicated than a typical parameterized actor network would allow. This is exactly the tradeoff explored in the graphical models literature between the use of Monte Carlo inference [11] and variational approximations [12].

Our algorithm is also related to probability matching [13], in which good actions are made more probable under the model, and the temperature at which the probability is computed is slowly reduced over time in order to move from exploration to exploitation and avoid local minima. Unlike our algorithm, the probability matching algorithm used a parameterized distribution which was maximized using gradient descent, and it did not address temporal credit assignment.

# 5  Conclusions

We have shown that a product of experts network can be used to learn the values of state-action pairs (including temporal credit assignment) when both the states and actions have a factored representation. An unbiased sample of actions can then be recovered with Gibbs sampling and 50 iterations appear to be sufficient. The network performs as well as a table-based Q-learner for small tasks, and continues to perform well when the task becomes too large for a table-based representation.

### Acknowledgments

We thank Peter Dayan, Zoubin Ghahramani and Andy Brown for helpful discussions. This research was funded by NSERC Canada and the Gatsby Charitable Foundation.

# References

[1] R.S Sutton and A.G. Barto. *Reinforcement Learning: An Introduction*. MIT Press, Cambridge, MA, 1998.

[2] A. G. Barto, R. S. Sutton, and C. W. Anderson. Neuronlike adaptive elements that can solve difficult learning control problems. *IEEE Transactions on Systems, Man and Cybernetics*, 13:835–846, 1983.

[3] R. S. Sutton. Integrated architectures for learning, planning, and reacting based on approximating dynamic programming. In *Proc. International Conference on Machine Learning*, 1990.

[4] Tommi Jaakkola, Satinder P. Singh, and Michael I. Jordan. Reinforcement learning algorithm for partially observable Markov decision problems. In Gerald Tesauro, David S. Touretzky, and Todd K. Leen, editors, *Advances in Neural Information Processing Systems*, volume 7, pages 345–352. The MIT Press, Cambridge, 1995.

[5] G. E. Hinton. Training products of experts by minimizing contrastive divergence. Technical Report GCNU TR 2000-004, Gatsby Computational Neuroscience Unit, UCL, 2000.

[6] S. Geman and D. Geman. Stochastic relaxation, Gibbs distributions, and the Bayesian restoration of images. *IEEE Transactions on Pattern Analysis and Machine Intelligence*, 6:721–741, 1984.

[7] G.A. Rummery and M. Niranjan. On-line Q-learning using connectionist systems. Technical Report CUED/F-INFENG/TR 166, Engineering Department, Cambridge University, 1994.

[8] R.S. Sutton. Generalization in reinforcement learning: Successful examples using sparse coarse coding. In Touretzky et al. [14], pages 1038–1044.

[9] M.L. Littman, A.R. Cassandra, and L.P. Kaelbling. Learning policies for partially observable environments: Scaling up. In *Proc. International Conference on Machine Learning*, 1995.

[10] D.P. Bertsekas and J.N. Tsitsiklis. *Neuro-Dynamic Programming*. Athena Scientific, Belmont, MA, 1996.

[11] R. M. Neal. Connectionist learning of belief networks. *Artificial Intelligence*, 56:71–113, 1992.

[12] T. S. Jaakkola. *Variational Methods for Inference and Estimation in Graphical Models*. Department of Brain and Cognitive Sciences, MIT, Cambridge, MA, 1997. Ph.D. thesis.

[13] Philip N. Sabes and Michael I. Jordan. Reinforcement learning by probability matching. In Touretzky et al. [14], pages 1080–1086.

[14] David S. Touretzky, Michael C. Mozer, and Michael E. Hasselmo, editors. *Advances in Neural Information Processing Systems*, volume 8. The MIT Press, Cambridge, 1996.
